# Learning Aspect Graph Representations from View Sequences

Michael Seibert and Allen M. Waxman
Lincoln Laboratory, Massachusetts Institute of Technology
Lexington, MA 02173-9108

## ABSTRACT

In our effort to develop a modular neural system for invariant learning and recognition of 3D objects, we introduce here a new module architecture called an *aspect network* constructed around adaptive axo-axo-dendritic synapses. This builds upon our existing system (Seibert & Waxman, 1989) which processes 2D shapes and classifies them into view categories (i.e., *aspects*) invariant to illumination, position, orientation, scale, and projective deformations. From a sequence of views, the aspect network learns the transitions between these aspects, crystallizing a graph-like structure from an initially amorphous network. Object recognition emerges by accumulating evidence over multiple views which activate competing object hypotheses.

## 1   INTRODUCTION

One can "learn" a three-dimensional object by exploring it and noticing how its appearance changes. When moving from one view to another, intermediate views are presented. The imagery is continuous, unless some feature of the object appears or disappears at the object's "horizon" (called the occluding contour). Such *visual events* can be used to partition continuously varying input imagery into a discrete sequence of aspects. The sequence of aspects (and the transitions between them) can be coded and organized into a representation of the 3D object under consideration. This is the form of 3D object representation that is learned by our *aspect network*. We call it an *aspect network* because it was inspired by the aspect graph concept of Koenderink and van Doorn (1979). This paper introduces this new network

which learns and recognizes sequences of aspects, and leaves most of the discussion of the visual preprocessing to earlier papers (Seibert & Waxman, 1989; Waxman, Seibert, Cunningham, & Wu, 1989). Presented in this way, we hope that our ideas of sequence learning, representation, and recognition are also useful to investigators concerned with speech, finite-state machines, planning, and control.

## 1.1  2D VISION BEFORE 3D VISION

The aspect network is one module of a more complete vision system (Figure 1) introduced by us (Seibert & Waxman, 1989). The early stages of the complete system learn and recognize 2D views of objects, invariant to the scene illumina-

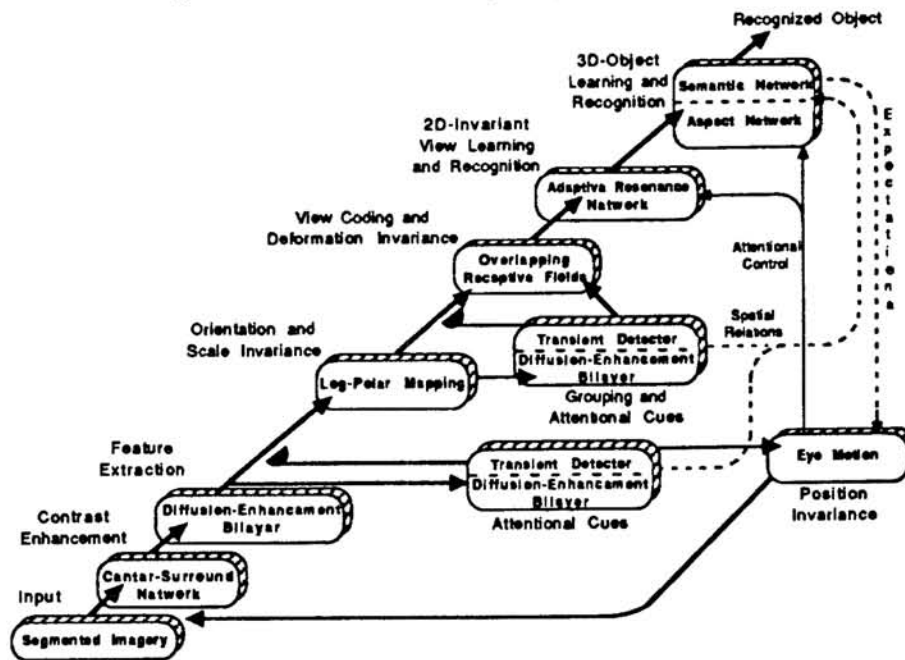

**Figure 1:** *Neural system architecture for 3D object learning and recognition.* The aspect network is part of the upper-right module.

tion and an object's orientation, size, and position in the visual field. Additionally, projective deformations such as foreshortening and perspective effects are removed from the learned 2D representations. These processing steps make use of Diffusion-Enhancement Bilayers (DEBs)[1] to generate attentional cues and featural groupings. The point of our neural preprocessing is to generate a sequence of views (i.e., aspects) which depends on the object's orientation in 3-space, but which does not depend on how the 2D images happen to fall on the retina. If no preprocessing were done, then the 3D representation would have to account for every possible 2D appearance in addition to the 3D information which relates the views to each other. Compressing the views into aspects avoids such combinatorial problems, but may result in an ambiguous representation, in that some aspects may be common to a number of objects. Such ambiguity is overcome by learning and recognizing a

*sequence* of aspects (i.e., a *trajectory* through the aspect graph). The partitioning and sequence recognition is analogous to building a symbol alphabet and learning syntactic structures within the alphabet. Each symbol represents an aspect and is encoded in our system as a separate category by an Adaptive Resonance Network architecture (Carpenter & Grossberg, 1987). This unsupervised learning is competitive and may proceed on-line with recognition; no separate training is required.

## 1.2   ASPECT GRAPHS AND OBJECT REPRESENTATIONS

Figure 2 shows a simplified aspect graph for a prismatic object.[2] Each node of

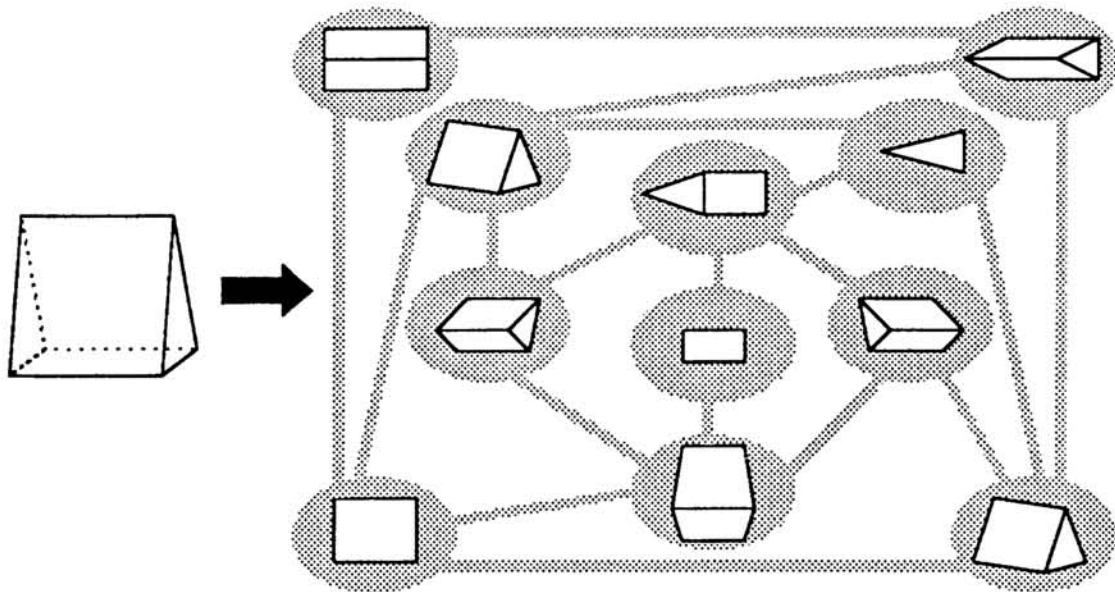

**Figure 2:** *Aspect Graph.* A 3D object can be represented as a graph of the characteristic view-nodes with adjacent views encoded by arcs between the nodes.

the graph represents a characteristic view, while the allowable transitions among views are represented by the arcs between the nodes. In this depiction, symmetries have been considered to simplify the graph. Although Koenderink and van Doorn suggested assigning aspects based on topological equivalences, we instead allow the ART 2 portion of our 2D system to decide when an invariant 2D view is sufficiently different from previously experienced views to allocate a new view category (aspect).

Transitions between adjacent aspects provide the key to the aspect network representation and recognition processes. Storing the transitions in a self-organizing synaptic weight array becomes the learned view-based representation of a 3D object. Transitions are exploited again during recognition to distinguish among objects with similar views. Whereas most investigators are interested in the computational complexity of generating aspect graphs from CAD libraries (Bowyer, Eggert, Stewman,

& Stark, 1989), we are interested in designing it as a self-organizing representation, learned from visual experience and useful for object recognition.

## 2    ASPECT-NETWORK LEARNING

The view-category nodes of ART 2 excite the aspect nodes (which we also call the $x$-nodes) of the aspect network (Figure 3). The aspect nodes fan-out to the dendritic

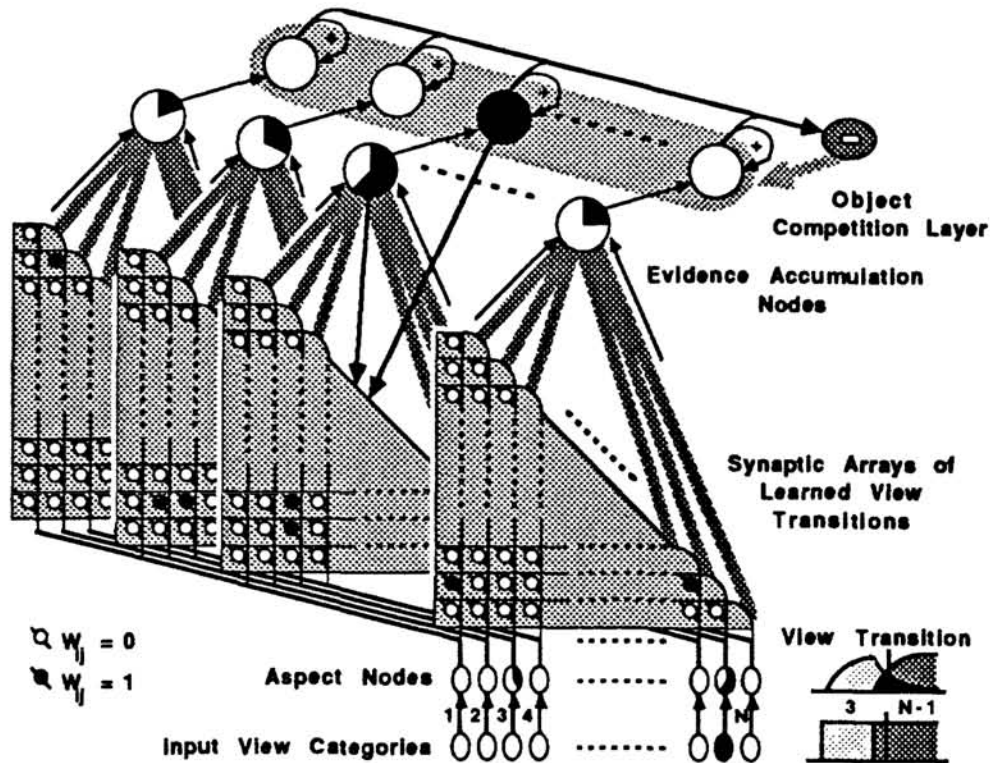

**Figure 3:** *Aspect Network.* The learned graph representations of 3D objects are realized as weights in the synaptic arrays. Evidence for experienced view-trajectories is simultaneously accumulated for all competing objects.

trees of *object neurons*. An object neuron consists of an adaptive synaptic array and an evidence accumulating $y$-node. Each object is learned by a single object neuron. A view sequence leads to accumulating activity in the $y$-nodes, which compete to determine the "recognized object" (i.e., maximally active $z$-node) in the "object competition layer". Gating signals from these nodes then modulate learning in the corresponding synaptic array, as in competitive learning paradigms. The system is designed so that the learning phase is integral with recognition. Learning (and forgetting) is always possible so that existing representations can always be elaborated with new information as it becomes available.

Differential equations govern the dynamics and architecture of the aspect network. These shunting equations model cell membrane and synapse dynamics as pioneered by Grossberg (1973, 1989). Input activities to the network are given by equation (1), the learned aspect transitions by equation (2), and the objects recognized from the experienced view sequences by equation (3).

## 2.1  ASPECT NODE DYNAMICS

The aspect node activities are governed by equation (1):

$$\frac{dx_i}{dt} \equiv \dot{x}_i = I_i - \lambda_x x_i, \tag{1}$$

where $\lambda_x$ is a passive decay rate, and $I_i = 1$ during the presentation of aspect $i$ and zero otherwise as determined by the output of the ART 2 module in the complete system (Figure 1). This equation assures that the activities of the aspect nodes build and decay in nonzero time (see the timetraces for the input $I$-nodes and aspect $x$-nodes in Figure 3). Whenever an aspect transition occurs, the activity of the previous aspect decays (with rate $\lambda_x$) and the activity of the new aspect builds (again with rate $\lambda_x$ in this case, which is convenient but not necessary). During the transient time when both activities are nonzero, only the synapses between these nodes have both pre- and post-synaptic activities which are significant (i.e., above the threshold) and Hebbian learning can be supported. The overlap of the pre- and post-synaptic activities is transient, and the extent of the transient is controlled by the selection of $\lambda_x$. This is the fundamental parameter for the dynamical behavior of the entire network, since it defines the response time of the aspect nodes to their inputs. As such, nearly every other parameter of the network depends on it.

## 2.2  VIEW TRANSITION ENCODING BY ADAPTIVE SYNAPSES

The aspect transitions that represent objects are realized by synaptic weights on the dendrite trees of object neurons. Equation (2) defines how the (initially small and random) weight relating aspect $i$, aspect $j$, and object $k$ changes:

$$\frac{dw_{ij}^k}{dt} \equiv \dot{w}_{ij}^k = \kappa_w\, w_{ij}^k\, (1 - w_{ij}^k)\, \{\Phi_w\,[(x_i + \epsilon)(x_j + \epsilon)] - \lambda_w\}\, \Theta_y(\dot{y}_k)\Theta_z(z_k). \tag{2}$$

Here, $\kappa_w$ governs the rate of evolution of the weights relative to the $x$-node dynamics, and $\lambda_w$ is the decay rate of the weights. Note that a small "background level" of activity $\epsilon$ is added to each $x$-node activity. This will be discussed in connection with (3) below. $\Phi_\phi(\gamma)$ is a threshold-linear function; that is: $\Phi_\phi(\gamma) = \gamma$ if $\gamma > \phi_{th}$ and zero otherwise. $\Theta_\theta(\gamma)$ is a binary-threshold function of the absolute-value of $\gamma$; that is: $\Theta_\theta(\gamma) = 1.0$ if $|\gamma| > \theta_{th}$ and zero otherwise.

Although this equation appears formidable, it can be understood as follows. Whenever simultaneous above-threshold activities arise presynaptically at node $x_i$ and postsynaptically at node $x_j$, the Hebbian product $(x_i + \epsilon)(x_j + \epsilon)$ causes $\dot{w}_{ij}^k$ to be positive (since above threshold, $(x_i + \epsilon)(x_j + \epsilon) > \lambda_w$) and the weight $w_{ij}^k$ learns the transition between the aspects $x_i$ and $x_j$. By symmetry, $w_{ji}^k$ would also learn, but all other weights decay ($\dot{w} \propto -\lambda_w$). The product of the shunting terms $w_{ij}^k(1 - w_{ij}^k)$ goes to zero (and thus inhibits further weight changes) only when $w_{ij}^k$ approaches either zero or unity. This shunting mechanism limits the range of weights, but also assures that these fixed points are invariant to input-activity magnitudes, decay-rates, or the initial and final network sizes.

The gating terms $\Theta_y(\dot{y}_k)$ and $\Theta_z(z_k)$ modulate the learning of the synaptic arrays $w_{ij}^k$. As a result of competition between multiple object hypotheses (see equation (4) below), only one $z_k$-node is active at a time. This implies recognition (or initial object neuron assignment) of "Object-$k$," and so only the synaptic array of Object-$k$ adapts. All other synaptic arrays $w_{ij}^l$ ($l \neq k$) remain unchanged. Moreover, learning occurs only during aspect transitions. While $\dot{y}_k \neq 0$ both learning and forgetting proceed; but while $\dot{y}_k \approx 0$ adaptation ceases though recognition continues (e.g. during a long sustained view).

## 2.3 OBJECT RECOGNITION DYNAMICS

Object nodes $y_k$ accumulate evidence over time. Their dynamics are governed by:

$$\frac{dy_k}{dt} \equiv \dot{y}_k = \kappa_y \left\{ \left[ \sum_i \sum_{j>i} \Phi_y \left[ (x_i + \epsilon)\, w_{ij}^k\, (x_j + \epsilon) \right] \right] - \lambda_y y_k \right\}. \quad (3)$$

Here, $\kappa_y$ governs the rate of evolution of the object nodes relative to the $x$-node dynamics, $\lambda_y$ is the passive decay rate of the object nodes, $\Phi_y(\cdot)$ is a threshold-linear function, and $\epsilon$ is the same small positive constant as in (2). The same Hebbian-like product (i.e., $(x_i+\epsilon)(x_j+\epsilon)$) used to learn transitions in (2) is used to detect aspect transitions during recognition in (3) with the addition of the synaptic term $w_{ij}^k$, which produces an axo-axo-dendritic synapse (see Section 3). Using this synapse, an aspect transition must not only be detected, but it must also be a permitted one for Object-$k$ (i.e., $w_{ij}^k > 0$) if it is to contribute activity to the $y_k$-node.

## 2.4 SELECTING THE MAXIMALLY ACTIVATED OBJECT

A "winner-take-all" competition is used to select the maximally active object node. The activity of each evidence accumulation $y$-node is periodically sampled by a corresponding object competition $z$-node (see Figure 3). The sampled activities then compete according to Grossberg's shunted short-term memory model (Grossberg, 1973), leaving only one $z$-node active at the expense of the activities of the other $z$-nodes. In addition to signifying the 'recognized' object, outputs of the $z$-nodes are used to inhibit weight adaptation of those weights which are not associated with the winning object via the $\Theta_z(z_k)$ term in equation (2). The competition is given by a first-order differential equation taken from (Grossberg, 1973):

$$\frac{dz_k}{dt} \equiv \dot{z}_k = \kappa_z \left[ f(z_k) - z_k \{ \lambda_z + \sum_l f(z_l) \} \right]. \quad (4)$$

The function $f(z)$ is chosen to be faster-than-linear (e.g. quadratic). The initial conditions are reset periodically to $z_k(0) = y_k(t)$.

## 3  THE AXO-AXO-DENDRITIC SYNAPSE

Although the learning is very closely Hebbian, the network requires a synapse that is more complex than that typically analyzed in the current modeling literature.

Instead of an axo-dendritic synapse, we utilize an *axo-axo-dendritic* synapse (Shepard, 1979). Figure 4 illustrates the synaptic anatomy and our functional model. We interpret the structure by assuming that it is the conjunction of activities in

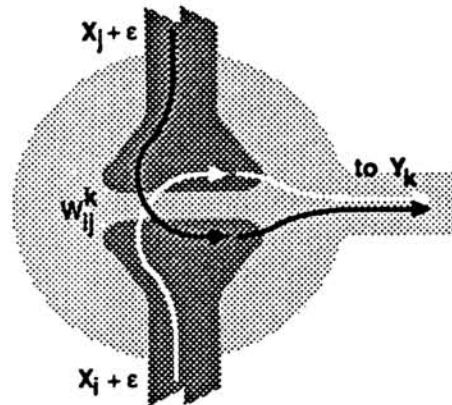

**Figure 4:** *Axo-axo-dendritic Synapse Model.* The Hebbian-like $w_{ij}^k$-weight adapts when simultaneous axonal activities $x_i$ and $x_j$ arise. Similarly, a conjunction of both activities is necessary to significantly stimulate the dendrite to node $y_k$.

both axons (as during an aspect transition) that best stimulates the dendrite. If, however, significant activity is present on only one axon (a sustained static view), it can stimulate the dendrite to a small extent in conjunction with the small base-level activity $\epsilon$ present on all axons. This property supports object recognition in static scenes, though object learning requires dynamic scenes.

## 4    SAMPLE RESULTS

Consider two objects composed of three aspects each with one aspect in common: the first has aspects 0, 2, and 4, while the second has aspects 0, 1, and 3. Figure 5 shows the evolution of the node activities and some of the weights during two aspect sequences. With an initial distribution of small, random weights, we present the repetitive aspect sequence $4 \rightarrow 2 \rightarrow 0 \rightarrow \cdots$, and learning is engaged by Object-1. The attention of the system is then redirected with a saccadic eye motion (the short-term memory node activities are reset to zero) and a new repetitive aspect sequence is presented: $3 \rightarrow 1 \rightarrow 0 \rightarrow \cdots$. Since the weights for these aspect transitions in the Object-1 synaptic array decayed as it learned its sequence, it does not respond strongly to this new sequence and Object-2 wins the competition. Thus, the second sequence is learned (and recognized!) by Object-2's synaptic weight array. In these simulations (1) - (4) were implemented by a Runge-Kutta coupled differential equation integrator. Each aspect was presented for $T = 4$ time-units. The equation parameters were set as follows: $I = 1$, $\lambda_x \approx \ln(0.1)/T$, $\lambda_y \approx 0.3$, $\lambda_w \approx 0.02$, $\kappa_y \approx 0.3$, $\kappa_w \approx 0.6$, $\epsilon \approx 0.03$, and thresholds of $\theta_y \approx 10^{-5}$ for $\Theta_y(\dot{y}_k)$ in equation (2), $\theta_z \approx 10^{-5}$ for $\Theta_z(z_k)$ in equation (2), $\phi_y > \epsilon^2$ for $\Phi_y$ in equation (3), $\phi_w > \max[\epsilon I/\lambda_x + \epsilon^2, (I/\lambda_x)^2 \exp(-\lambda_x T)]$ for $\Phi_w$ in equation (2). The $\phi_w$ constraint insures that only transitions are learned, and they are learned only when $t < T$.

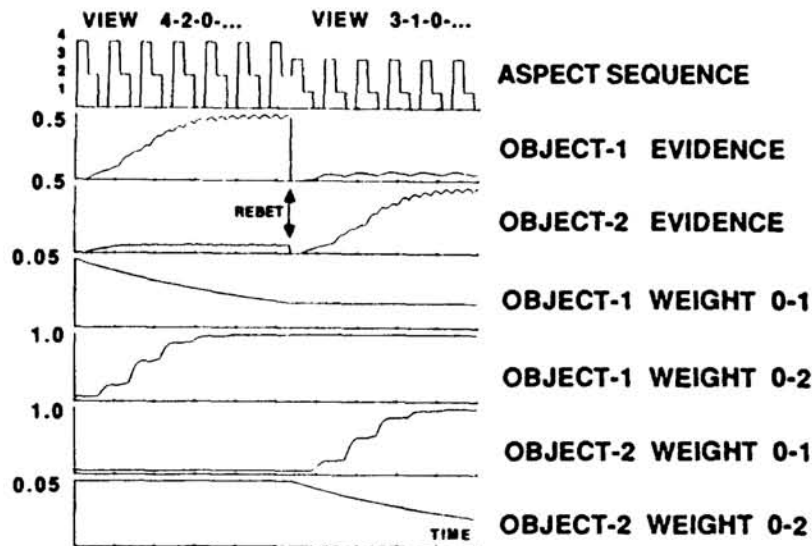

**Figure 5:** *Node activity and synapse adaptation vs. time.* Two separate representations are learned automatically as aspect sequences of the objects are experienced.

## Acknowledgments

This report is based on studies performed at Lincoln Laboratory, a center for research operated by the Massachusetts Institute of Technology. The work was sponsored by the Department of the Air Force under Contract F19628-85-C-0002.

## Footnotes

[1]This architecture was previously called the NADEL (Neural Analog Diffusion-Enhancement Layer), but has been renamed to avoid causing any problems or confusion, since there is an active researcher in the field with this name.

[2]Neither the aspect graph concept nor our aspect network implementation is limited to simple polyhedral objects, nor must the objects even be convex, i.e., they may be self-occluding.

## References

Bowyer, K., Eggert, D., Stewman, J., & Stark, L. (1989). Developing the aspect graph representation for use in image understanding. *Proceedings of the 1989 Image Understanding Workshop.* Wash. DC: DARPA. 831-849.

Carpenter, G. A., & Grossberg, S. (1987). ART 2: Self-organization of stable category recognition codes for analog input patterns. *Applied Optics,* **26**(23), 4919-4930.

Grossberg, S. (1973). Contour enhancement, short term memory, and constancies in reverberating neural networks. *Studies in Applied Mathematics,* **52**(3), 217-257.

Koenderink, J. J., & van Doorn, A. J. (1979). The internal representation of solid shape with respect to vision. *Biological Cybernetics,* **32**, 211-216.

Seibert, M., Waxman, A. M. (1989). Spreading Activation Layers, Visual Saccades, and Invariant Representations for Neural Pattern Recognition Systems. *Neural Networks.* 2(1). 9-27.

Shepard, G. M. (1979). **The synaptic organization of the brain.** New York: Oxford University Press.

Waxman, A. M., Seibert, M., Cunningham, R., & Wu, J. (1989). Neural analog diffusion-enhancement layer and spatio-temporal grouping in early vision. In: **Advances in neural information processing systems,** D. S. Touretzky (ed.), San Mateo, CA: Morgan Kaufman. 289-296.